# Support Vector Method for Multivariate Density Estimation

**Vladimir N. Vapnik**
Royal Halloway College and
AT&T Labs, 100 Schultz Dr.
Red Bank, NJ 07701
*vlad@research.att.com*

**Sayan Mukherjee**
CBCL, MIT E25-201
Cambridge, MA 02142
*sayan@ai.mit.edu*

## Abstract

A new method for multivariate density estimation is developed based on the Support Vector Method (SVM) solution of inverse ill-posed problems. The solution has the form of a mixture of densities. This method with Gaussian kernels compared favorably to both Parzen's method and the Gaussian Mixture Model method. For synthetic data we achieve more accurate estimates for densities of 2, 6, 12, and 40 dimensions.

## 1  Introduction

The problem of multivariate density estimation is important for many applications, in particular, for speech recognition [1] [7]. When the unknown density belongs to a parametric set satisfying certain conditions one can estimate it using the maximum likelihood (ML) method. Often these conditions are too restrictive. Therefore, non-parametric methods were proposed.

The most popular of these, Parzen's method [5], uses the following estimate given data $x_1, ..., x_\ell$:

$$P(t, \gamma_\ell) = \frac{1}{\ell} \sum_{i=1}^{\ell} K_{\gamma_\ell}(t - x_i), \tag{1}$$

where $K_{\gamma_\ell}(t - x_i)$ is a smooth function such that $\int K_{\gamma_\ell}(t - x_i)dt = 1$. Under some conditions on $\gamma_\ell$ and $K_{\gamma_\ell}(t - x_i)$, Parzen's method converges with a fast asymptotic rate. An example of such a function is a Gaussian with one free parameter $\gamma_\ell^2$ (the width)

$$K_{\gamma_\ell}(\mathbf{t} - \mathbf{x}_i) = \frac{1}{(2\pi)^{n/2}\gamma_\ell^n} \exp\left\{-(\mathbf{t} - \mathbf{x}_i)^T \left(\gamma_\ell^2 I\right)^{-1} (\mathbf{t} - \mathbf{x}_i)\right\}. \tag{2}$$

The structure of the Parzen estimator is too complex: the number of terms in (1) is equal to the number of observations (which can be hundreds of thousands).

Researchers believe that for practical problems densities can be approximated by a mixture with few elements (Gaussians for Gaussian Mixture Models (GMM)). Therefore, the following parametric density model was introduced

$$P(\mathbf{x}, \mathbf{a}, \Sigma) = \sum_{i=1}^{m} \alpha_i P(\mathbf{x}, \mathbf{a}_i, \Sigma_i), \quad \alpha \geq 0, \ \sum_{i=1}^{m} \alpha_i = 1, \qquad (3)$$

where $P(\mathbf{x}, \mathbf{a}_i, \Sigma_i)$ are Gaussians with different vectors $\mathbf{a}_i$ and different diagonal covariance matrices $\Sigma_i$; $\alpha_i$ is the proportion of the $i$-th Gaussian in the mixture.

It is known [9] that for general forms of Gaussian mixtures the ML estimate does not exist. To use the ML method two values are specified: a lower bound on diagonal elements of the covariance matrix and an upper bound on the number of mixture elements. Under these constraints one can estimate the mixture parameters using the EM algorithm. This solution, however, is based on predefined parameters.

In this article we use an SVM approach to obtain an estimate in the form of a mixture of densities. The approach has no free parameters. In our experiments it performs better than the GMM method.

## 2  Density estimation is an ill-posed problem

A density $p(t)$ is defined as the solution of the equation

$$\int_{-\infty}^{x} p(t)\,dt = F(x), \qquad (4)$$

where $F(x)$ is the probability distribution function. Estimating a density from data involves solving equation (4) on a given set of densities when the distribution function $F(x)$ is unknown but a random i.i.d. sample $x_1, ..., x_\ell$ is given. The empirical distribution function $F_\ell(x)$ is a good approximation of the actual distribution,

$$F_\ell(x) = \frac{1}{\ell} \sum_{i=1}^{\ell} \theta(x - x_i),$$

where $\theta(u)$ is the step-function. In the univariate case, for sufficiently large $\ell$ the distribution of the supremum error between $F(x)$ and $F_\ell(x)$ is given by the Kolmogorov-Smirnov distribution

$$P\{\sup_x |F(x) - F_\ell(x)| < \varepsilon/\sqrt{\ell}\} = 1 - 2\sum_{k=1}^{\infty} (-1)^{k-1} \exp\{-2\varepsilon^2 k^2\}. \qquad (5)$$

Hence, the problem of density estimation can be restated as solving equation (4) but replacing the distribution function $F(x)$ with the empirical distribution function $F_\ell(x)$ which converges to the true one with the (fast) rate $O(\frac{1}{\sqrt{\ell}})$, for univariate and multivariate cases.

The problem of solving the linear operator equation $Ap = F$ with approximation $F_\ell(x)$ is ill-posed.

In the 1960's methods were proposed for solving ill-posed problems using approximations $F_\ell$ converging to $F$ as $\ell$ increases. The idea of these methods was to

introduce a regularizing functional $\Omega(p)$ (a semi-continuous, positive functional for which $\Omega(p) \leq c$, $c > 0$ is a compactum) and define the solution $p_\ell$ which is a trade-off between $\Omega(p)$ and $||Ap - F_\ell||$.

The following two methods which are asymptotically equivalent [11] were proposed by Tikhonov [8] and Phillips [6]

$$\min_p \left[ ||Ap - F_\ell||^2 + \gamma_\ell \Omega(p) \right], \quad \gamma_\ell > 0, \quad \gamma_\ell \to 0, \tag{6}$$

$$\min_p \Omega(p) \quad s.t. \quad ||Ap - F_\ell|| < \varepsilon_\ell, \quad \varepsilon_\ell > 0, \quad \varepsilon_\ell \to 0. \tag{7}$$

For the stochastic case it can be shown for both methods that if $F_\ell(x)$ converges in probability to $F(x)$ and $\gamma_\ell \to 0$ then for sufficiently large $\ell$ and arbitrary $\nu$ and $\mu$ the following inequality holds [10] [9] [3]

$$P(\rho_{E_1}(p, p_\ell) > \nu) \leq P(\rho_{E_2}(F, F_\ell) > \sqrt{\gamma_\ell \mu}) \tag{8}$$

where $\ell > \ell_0(\nu, \mu)$ and $\rho_{E_1}(p, p_\ell)$, $\rho_{E_2}(F, F_\ell)$ are metrics in the spaces $p$ and $F$. Since $F_\ell(x) \to F(x)$ in probability with the rate $O(\frac{1}{\sqrt{\ell}})$, from equation (8) it follows that if $\gamma_\ell > O(\frac{1}{\sqrt{\ell}})$ the solutions of equation (4) are consistent.

## 3 Choice of regularization parameters

For the deterministic case the residual method [2] can be used to set the regularization parameters ($\gamma_\ell$ in (6) and $\varepsilon_\ell$ in (7)) by setting the parameter ($\gamma_\ell$ or $\varepsilon_\ell$) such that $p_\ell$ satisfies the following

$$||Ap_\ell - F_\ell|| = ||F(x) - F_\ell(x)|| = \sigma_\ell, \tag{9}$$

where $\sigma_\ell$ is the known accuracy of approximation of $F(x)$ by $F_\ell(x)$. We use this idea for the stochastic case. The Kolmogorov-Smirnov distribution is used to set $\sigma_\ell$, $\sigma_\ell = c/\sqrt{\ell}$, where $c$ corresponds to an appropriate quantile. For the multivariate case one can either evaluate the appropriate quantile analytically [4] or by simulations.

The density estimation problem can be solved using either regularization method (6) or (7). Using method (6) with a $L_2$ norm in image space $F$ and regularization functional $\Omega(p) = (\mathcal{T}p, \mathcal{T}p)$ where $\mathcal{T}$ is a convolution operator, one obtains Parzen's method [10] [9] with kernels defined by operator $\mathcal{T}$.

## 4 SVM for density estimation

We apply the SVM technique to equation (7) for density estimation. We use the $C$ norm in (7) and solve equation (4) in a set of functions belonging to a Reproducing Kernel Hilbert Space (RKHS). We use the regularization functional

$$\Omega(p) = ||p||_{\mathcal{H}}^2 = (p, p)_{\mathcal{H}}. \tag{10}$$

A RKHS can be defined by a positive definite kernel $K(x, y)$ and an inner product $(f, g)_{\mathcal{H}}$ in Hilbert space $\mathcal{H}$ such that

$$(f(x), K(x, y))_{\mathcal{H}} = f(y) \quad \forall f \in \mathcal{H}. \tag{11}$$

Note that any positive definite function $K(x, y)$ has an expansion

$$K(x, y) = \sum_{i=1}^{\infty} \lambda_i \phi_i(x) \phi_i(y) \tag{12}$$

where $\lambda_i$ and $\phi_i(x)$ are eigenvalues and eigenfunctions of $K(x, y)$. Consider the set of functions

$$f(x, c) = \sum_{i=1}^{\infty} c_i \phi_i(x) \tag{13}$$

and the inner product

$$(f(x, c^*), f(x, c^{**})) = \sum_{i=1}^{\infty} \frac{c_i^* c_i^{**}}{\lambda_i}. \tag{14}$$

Kernel (12), inner product (14), and set (13) define a RKHS and

$$(f(x), K(x, y))_{\mathcal{H}} = \left( \sum_{i=1}^{\infty} c_i \phi_i(x), K(x, y) \right)_{\mathcal{H}} =$$

$$\left( \sum_{i=1}^{\infty} c_i \phi_i(x), \sum_{i=1}^{\infty} \lambda_i \phi_i(x) \phi_i(y) \right)_{\mathcal{H}} = \sum_{i=1}^{\infty} \frac{c_i \lambda_i \phi_i(y)}{\lambda_i} = f(y).$$

For functions from a RKHS the functional (10) has the form

$$\Omega(p) = \sum_{i=1}^{\infty} \frac{c_i^2}{\lambda_i}, \tag{15}$$

where $\lambda_i$ is the $i$-th eigenvalue of the kernel $K(x, y)$. The choice of the kernel defines smoothness properties on the solution.

To use method (7) to solve for the density in equation (4) in a RKHS with a solution satisfying condition (9) we minimize

$$\Omega(p) = (p, p)_{\mathcal{H}}$$

subject to the constraint

$$\max_i \left| F_\ell(x) - \int_{-\infty}^{x} p(t) dt \right|_{x = x_i} = \sigma_\ell.$$

We look for a solution of equation (4) with the form

$$p(t) = \sum_{i=1}^{\ell} \beta_i K_{\gamma_\ell}(x_i, t). \tag{16}$$

Accounting for (16) and (11) minimizing (10) is equivalent to minimizing

$$\Omega(p, p) = (p, p)_{\mathcal{H}} = \sum_{i,j=1}^{\ell} \beta_i \beta_j K_{\gamma_\ell}(x_i, x_j) \tag{17}$$

subject to constraints

$$\max_i \left| F_\ell(x) - \sum_{j=1}^{\ell} \beta_j \int_{-\infty}^{x} K_{\gamma_\ell}(x_j, t)dt \right|_{x=x_i} = \sigma_\ell, \tag{18}$$

$$\beta_i \geq 0, \quad \sum_{i=1}^{\ell} \beta_i = 1. \tag{19}$$

This optimization problem is closely related to the SV regression problem with an $\sigma_\ell$-insensitive zone [9]. It can be solved using the standard SVM technique.

Generally, only a few of the $\beta_i$ will be nonzero, the $x_i$ corresponding to these $\beta_i$ are called support vectors.

Note that kernel (2) has width parameter $\gamma_\ell$. We call the value of this parameter admissible if it satisfies constraint (18) (the solution satisfies condition (9)). The admissible set $\gamma_{min} \leq \gamma_\ell \leq \gamma_{max}$ is not empty since for Parzen's method (which also has form (16)) such a value does exist. Among the $\gamma_\ell$ in this admissible set we select the one for which $\Omega(p)$ is smallest or the number of support vectors is minimum.

Choosing other kernels (for example Laplacians) one can estimate densities using non-Gaussian mixture models which for some problems are more appropriate [1].

## 5 Experiments

Several trials of estimates constructed from sampling distributions were examined. Boxplots were made of the $L_1(p)$ norm over the trials. The horizontal lines of the boxplot indicate the 5%, 25%, 50%, 75%, and 95% quantiles of the error distribution.

For the SVM method we set $\sigma_\ell = c/\sqrt{\ell}$, where $c = .36, .41, .936$, and $1.75$ for two, six, twelve and forty dimensions. For Parzen's method $\gamma_\ell$ was selected using a leave-one out procedure. The GMM method uses the EM algorithm and sets all parameters except $n$, the upper bound on the number of terms in the mixture [7].

Figure (1) shows plots of the SVM estimate using a Gaussian kernel and the GMM estimate when 60 points were drawn form a mixture of a Gaussian and Laplacian in two dimensions.

Figure (2a) shows four boxplots of estimating a density defined by a mixture of two Laplacians in a two dimensional space using 200 observations. Each boxplot shows outcomes of 100 trials: for the SVM method, Parzen's method, and the GMM method with parameters $n = 2$, and $n = 4$. Figure (2c) shows the distribution of the number of terms for the SVM method.

Figure (2b) shows boxplots of estimating a density defined by the mixture of four Gaussians in a six dimensional space using 600 observations. Each boxplot shows outcomes of 50 trials: for the SVM method, Parzen's method, and the GMM method with parameters $n = 4$, and $n = 8$. Figure (2c) shows the distribution of the number of terms for the SVM method.

Figure (3a) shows boxplots of outcomes of estimating a density defined by the mixture of four Gaussians and four Laplacians in a twelve dimensional space using

400 observations. Each boxplot shows outcomes of 50 trials: for the SVM method, Parzen's method, and the GMM method with parameter $n = 8$. Figure (3c) shows the distribution of the number of terms for the SVM method.

Figure (3b) shows boxplots of outcomes of estimating a density defined by the mixture of four Gaussians and four Laplacians in a forty dimensional space using 480 observations. Each box-plot shows outcomes of 50 trials: for the SVM method, Parzen's method, and the GMM method with parameter $n = 8$. Figure (3c) shows the distribution of the number of terms for the SVM method.

## 6   Summary

A method for multivariate density estimation based on the SVM technique for solving ill-posed problems is introduced. This method has a form of a mixture of densities. The estimate in general has only a few terms. In experiments on synthetic data this method is more accurate than the GMM method.

## References

[1] S. Basu and C.A. Micchelli. Parametric density estimation for the classification of acoustic feature vectors in speech recognition. In *Nonlinear Modeling, Advanced Black-Box Techniques*. Kluwer Publishers, 1998.

[2] V.A. Morozov. *Methods for solving incorrectly posed problems*. Springer-Verlag, Berlin, 1984.

[3] S. Mukherjee and V. Vapnik. Multivariate density estimation: An svm approach. AI Memo 1653, Massachusetts Institute of Technology, 1999.

[4] S. Paramasamy. On multivariate kolmogorov-smirnov distribution. *Statistics & Probability Letters*, 15:140–155, 1992.

[5] E. Parzen. On estimation of a probability density function and mode. *Ann. Math. Statis.*, 33:1065–1076, 1962.

[6] D.L. Phillips. A technique for the numerical solution of integral equations of the first kind. *J.Assoc. Comput. Machinery*, 9:84–97, 1962.

[7] D. Reynolds and R. Rose. Robust text-independent speaker identification using gaussian mixture speaker models. *IEEE Trans on Speech and Audio Processing*, 3(1):1–27, 1995.

[8] A. N. Tikhonov. Solution of incorrectly formulated problems and the regularization method. *Soviet Math. Dokl.*, 4:1035–1038, 1963.

[9] V. N. Vapnik. *Statistical learning theory*. J. Wiley, 1998.

[10] V.N. Vapnik and A.R. Stefanyuk. Nonparametric methods for restoring probability densities. *Avtomatika i Telemekhanika*, (8):38–52, 1978.

[11] V.V. Vasin. Relationship of several variational methods for the approximate solution of ill-posed problems. *Math Notes*, 7:161–166, 1970.

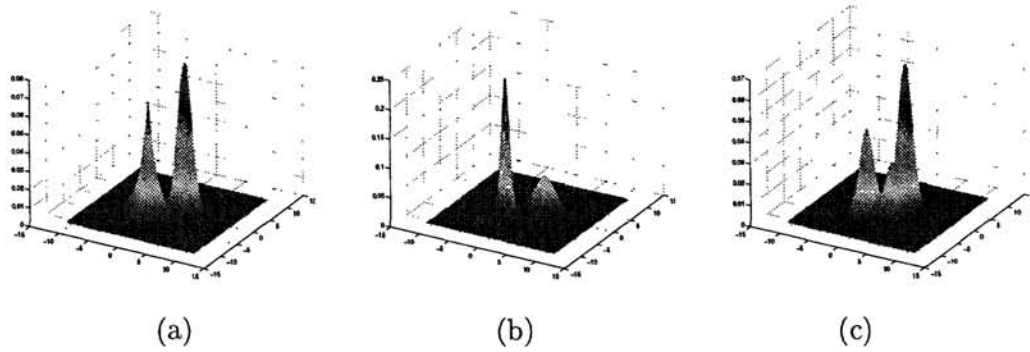

Figure 1: (a) The true distribution (b) the GMM case with 4 mixtures (c) the Parzen case (d) the SVM case for 60 points.

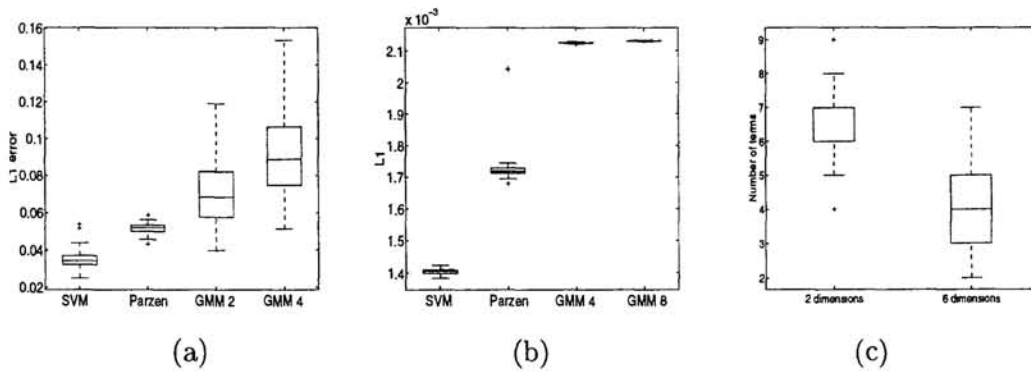

Figure 2: (a) Boxplots of the $L_1(p)$ error for the mixture of two Laplacians in two dimensions for the SVM method, Parzen's method, and the GMM method with 2 and 4 Gaussians. (b) Boxplots of the $L_1(p)$ error for mixture of four Gaussians in six dimensions with the SVM method, Parzen's method, and the GMM method with 4 mixtures. (c) Boxplots of distribution of the number of terms for the SVM method for the two and six dimensional cases.

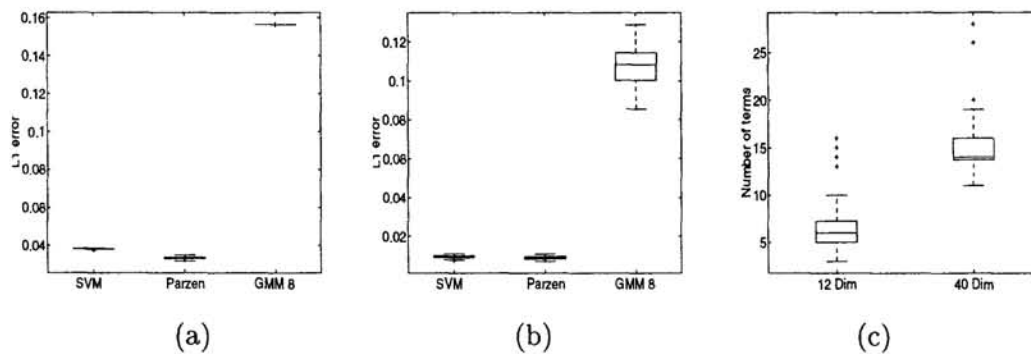

Figure 3: (a) Boxplots of the $L_1(p)$ error for the mixture of four Laplacians and four Gaussians in twelve dimensions for the SVM method, Parzen's method, and the GMM method with 8 Gaussians. (b) Boxplots of the $L_1(p)$ error for the mixture of four Laplacians and four Gaussians in forty dimensions for the SVM method, Parzen's method, and the GMM method with 8 Gaussians. (c) Boxplots of distribution of the number of terms for the SVM method for the twelve and forty dimensional cases.